# Extracting Relevant Structures with Side Information

**Gal Chechik** and **Naftali Tishby**
*{ggal,tishby}@cs.huji.ac.il*
School of Computer Science and Engineering and
The Interdisciplinary Center for Neural Computation
The Hebrew University of Jerusalem, 91904, Israel

## Abstract

The problem of extracting the relevant aspects of data, in face of multiple conflicting structures, is inherent to modeling of complex data. Extracting structure in one random variable that is relevant for another variable has been principally addressed recently via the *information bottleneck method* [15]. However, such auxiliary variables often contain more information than is actually required due to structures that are irrelevant for the task. In many other cases it is in fact easier to specify what is irrelevant than what is, for the task at hand. Identifying the relevant structures, however, can thus be considerably improved by also minimizing the information about another, irrelevant, variable. In this paper we give a general formulation of this problem and derive its formal, as well as algorithmic, solution. Its operation is demonstrated in a synthetic example and in two real world problems in the context of text categorization and face images. While the original information bottleneck problem is related to rate distortion theory, with the distortion measure replaced by the relevant information, extracting relevant features while removing irrelevant ones is related to rate distortion with side information.

## 1   Introduction

A fundamental goal of machine learning is to find regular structures in a given empirical data, and use it to construct predictive or comprehensible models. This general goal, unfortunately, is very ill defined, as many data sets contain alternative, often conflicting, underlying structures. For example, documents may be classified either by subject or by writing style; spoken words can be labeled by their meaning or by the identity of the speaker; proteins can be classified by their structure or function - all are valid alternatives. Which of these alternative structures is "relevant" is often implicit in the problem formulation.

The problem of identifying "the" relevant structures is commonly addressed in supervised learning tasks, by providing a "relevant" label to the data, and selecting features that are discriminative with respect to this label. An information theoretic generalization of this supervised approach has been proposed in [9, 15] through the *information bottleneck method* (IB). In this approach, relevance is introduced through another random variable (as is the label in supervised learning) and the goal is to compress one (the source) variable, while maintaining as much information about the auxiliary (relevance) variable. This framework

has proven powerful for numerous applications, such as clustering the objects of sentences with respect to the verbs [9], documents with respect to their terms [1, 6, 14], genes with respect to tissues [8, 11], and stimuli with respect to spike patterns [10].

An important condition for this approach to work is that the auxiliary variable indeed corresponds to the task. In many situations, however, such "pure" variable is not available. The variable may in fact contain alternative and even conflicting structures. In this paper we show that this general and common problem can be alleviated by providing "negative information", i.e. information about "unimportant", or irrelevant, aspects of the data that can interfere with the desired structure during the learning.

As an illustration, consider a simple nonlinear regression problem. Two variables $x$ and $y$ are related through a functional form $y = f(x) + \xi$, where $f(x)$ is in some known function class and $\xi$ is noise with some distribution that *depends on* $x$. When given a sample of $(x, y)$ pairs with the goal of extracting the relevant dependence $y = f(x)$, the noise $\xi$ - which may contain information on $x$ and thus interfere with extracting $y$ - is an irrelevant variable. Knowing the joint distribution of $(x, \xi)$ can of course improve the regression result.

A more "real life" example can be found in the analysis of gene expression data. Such data, as generated by the DNA-chips technology, can be considered as an empirical joint distribution of gene expression levels and different tissues, where the tissues are taken from different biological conditions and pathologies. The search for expressed genes that testify for the existence of a pathology may be obscured by genetic correlations that exist also in other conditions. Here again a sample of irrelevant expression data, taken for instance from a healthy population, can enable clustering analysis to focus on the pathological features only, and ignore spurious structures.

These two examples, and numerous others, are all instantiations of a common problem: in order to better extract the relevant structures information about the irrelevant components of the data should be incorporated. Naturally, various solutions have been suggested to this basic problem in many different contexts (e.g. spectral subtraction, weighted regression analysis). The current paper presents a general unified information theoretic framework for such problems, extending the original information bottleneck variational problem to deal with discriminative tasks of that nature, by observing its analogy with rate distortion theory with side information.

## 2 Information Theoretic Formulation

To formalize the problem of extracting relevant structures consider first three categorical variables $X$, $Y^+$ and $Y^-$ whose co-occurrence distributions are known. Our goal is to uncover structures in $P(X, Y^+)$, that do not exist in $P(X, Y^-)$. The distribution $P(X; Y^+)$ may contain several conflicting underlying structures, some of which may also exist in $P(X, Y^-)$. These variables stand for example for a set of terms $X$, a set of documents $Y^+$ whose structure we seek, and an additional set of documents $Y^-$, or a set of genes and two sets of tissues with different biological conditions. In all these examples $Y^+$ and $Y^-$ are *conditionally independent* given $X$. We thus make the assumption that the joint distribution factorizes as: $p(x, y^+, y^-) = p(x)p(y^+|x)p(y^-|x)$.

The relationship between the variables can be expressed by a Venn diagram (Figure 1A), where the area of each circle corresponds to the entropy of a variable (see e.g. [2] p.20 and [3] p.50 for discussion of this type of diagrams) and the intersection of two circles corresponds to their mutual information. The mutual information of two random variables is the familiar symmetric functional of their joint distribution, $I(X; Y) = \sum_{x,y} p(x, y) \log \left( \frac{p(x,y)}{p(x)p(y)} \right)$ .

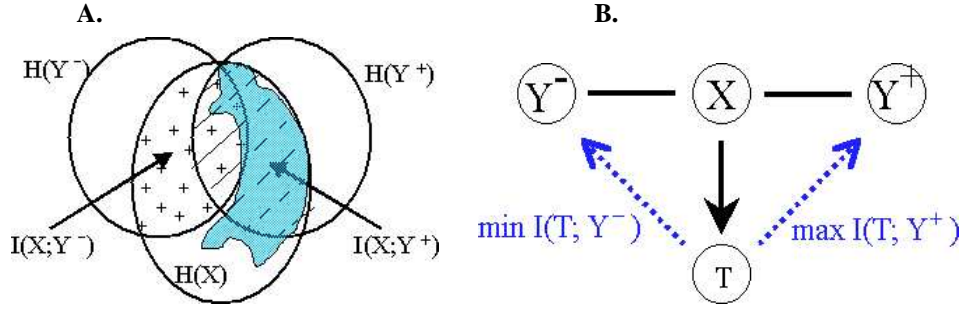

Figure 1: **A.** A Venn diagram illustrating the relations between the entropy and mutual information of the variables $X$,$Y^+$,$Y^-$. The area of each circle corresponds to the entropy of a variable, while the intersection of two circles corresponds to their mutual information. As $Y^+$ and $Y^-$ are independent given $X$, their mutual information vanishes when $x$ is known, thus all their overlap is included in the circle of $X$. **B.** A graphical model representation of IB with side information. Given the three variables $X$,$Y^+$,$Y^-$, we seek a compact stochastic representation $T$ of $X$ which preserves information about $Y^+$ but removes information about $+Y^-$. In this graph $Y^+$ and $Y^-$ are indeed conditionally independent given $X$.

To identify the relevant structures in the joint distribution $p(x, y^+)$, we aim to extract a compact representation of the variable $X$ with minimal loss of mutual information about the relevant variable $Y^+$, and *at the same time* with maximal loss of information about the irrelevance variable $Y^-$. The goal of *information bottleneck with side information* (IBSI) is therefor to find a stochastic map of $X$ to a new variable $T$, $p(t|x)$, in a way that maximizes its mutual information with $Y^+$ and minimizes the mutual information about $Y^-$. In general one can achieve this goal perfectly only asymptotically and the finite case leads to a sub optimal compression, an example of which is depicted in the blue region in figure 1. These constrains can be cast into a single variational functional,

$$\mathcal{L} = I(X;T) - \beta \left[ I(T;Y^+) - \gamma I(T;Y^-) \right] \tag{1}$$

where the Lagrange parameter $\beta$ determines the tradeoff between compression and information extraction while the parameter $\gamma$ determines the tradeoff between preservation of information about the relevant $Y^+$ variable and loss of information about the irrelevant one $Y^-$. In some applications, such as in communication, the value of $\gamma$ may be determined by the relative cost of transmitting the information about $Y^-$ by other means.

The *information bottleneck* variational problem, introduced in [15], is a special case of our current variational problem with $\gamma = 0$, namely, no side or irrelevant information is available. In that case only the distributions $p(t|x)$, $p(t)$ and $p(y^+|t)$ are determined.

## 3 Solution Characterization

The complete Lagrangian of this constrained optimization problem is given by

$$\mathcal{L}[p(t|x)] = I(X;T) - \beta \left[ I(T;Y^+) - \gamma I(T;Y^-) \right] - \sum_x \lambda(x) \sum_t p(t|x) \tag{2}$$

where $\lambda(x)$, are the normalization Lagrange multipliers. Here, the minimization is performed with respect to the stochastic mapping $p(t|x)$, taking into account its probabilistic relations to $p(t)$, $p(y^+|t)$ and $p(y^-)$. Interestingly, performing the minimization over $p(t|x); p(t); p(y^+|t); p(y^-|t)$ as independent variables leads to the same solution of self-consistent equations.

**Proposition 1** *The extrema of $\mathcal{L}$ obey the following self consistent equations*

$$p(t|x) = \frac{p(t)}{Z}e^{\left(-\beta\left(D_{KL}\left[p(y^+|x)||p(y^+|t)\right]-\gamma D_{KL}\left[p(y^-|x)||p(y^-|t)\right]\right)\right)} \qquad (3)$$

$$p(t) = \sum_x p(t|x)p(x)$$

$$p(y^+|t) = \frac{1}{p(t)}\sum_x p(y^+|x)p(t|x)p(x)$$

$$p(y^-|t) = \frac{1}{p(t)}\sum_x p(y^-|x)p(t|x)p(x)$$

*where $Z = p(t)\sum_t \exp\left\{-\beta\left(D_{KL}\left[p(y^+|x)||p(y^+|t)\right]-\gamma D_{KL}\left[p(y^-|x)||p(y^-|t)\right]\right)\right\}$ is a normalization factor and $D_{KL}\left[p||q\right] = \sum_x p(x)\log\frac{p(x)}{q(x)}$ is the Kullback-Leibler divergence [2],*

**Proof:**
Following the Markovian relation $p(y|x,t) = p(y|x)$, we write $p(y,t) = \sum_x p(y,t|x)p(x)$ $= \sum_x p(y|t,x)p(t|x)p(x) = \sum_x p(y|x)p(t|x)p(x)$ and obtain for the second term of Eq. 3

$$\frac{\delta}{\delta p(t|x)}I(T;Y^+) = \frac{\delta}{\delta p(t|x)}\sum_t\sum_{y^+}\sum_x p(y^+|x)p(t|x)p(x)\log\left(\frac{p(y^+|t)}{p(y^+)}\right) \qquad (4)$$

$$= p(x)\sum_{y^+}p(y^+|x)\log\left(\frac{p(y^+|t)}{p(y^+|x)}\frac{p(y^+|x)}{p(y^+)}\right)$$

$$= -p(x)D_{KL}\left[p(y^+|x)||p(y^+|t)\right] + p(x)D_{KL}\left[p(y^+|x)||p(y^+)\right]\ .$$

Similar differentiation for the other terms yield

$$\frac{\delta}{\delta p(t|x)}\mathcal{L} = p(x)\log\left(\frac{p(t|x)}{p(t)}\right) \qquad (5)$$

$$-p(x)\beta\left(D_{KL}\left[p(y^+|x)||p(y^+|t)\right] + \gamma D_{KL}\left[p(y^-|x)||p(y^-|t)\right]\right)$$

$$+p(x)\lambda(x,y^+,y^-)$$

where $\lambda(x,y^+,y^-) = \frac{\lambda(x)}{p(x)} + \beta\left(D_{KL}\left[p(y^+|x)||p(y^+)\right] - \gamma D_{KL}\left[p(y^-|x)||p(y^-)\right]\right)$,

holds all terms independent of $t$. Equating the derivative to zero then yields the first equation of proposition 1. $\square$

The formal solutions of the above variational problem have an exponential form which is a natural generalization of the solution of the original IB problem. As in the original IB, when $\beta$ goes to infinity the Lagrangian reduces to $I(T;Y^+) - \gamma I(T;Y^-)$, and the exponents collapse to a hard clustering solution, where $p(t|x)$ become binary cluster membership probabilities.

Further intuition about the operation of IBSI can be obtained by rewriting the second term in Eq. 2, $I(T;Y^+) - \gamma I(T;Y^-) = \sum_t\sum_{y^+}\sum_{y^-}p(t,y^+,y^-)\log\left(\frac{p(y^+|t)}{p(y^+)}\right) - \gamma\sum_t\sum_{y^+}\sum_{y^-}p(t,y^+,y^-)\log\left(\frac{p(y^-|t)}{p(y^-)}\right) = \left\langle\log\left(\frac{p(y^+|t)}{p(y^-|t)^\gamma}\frac{p(y^-)^\gamma}{p(y^+)}\right)\right\rangle_{p(t,y^+,y^-)}$. For $\gamma = 1$ and a fixed level of $I(X;T)$, IBSI thus operates to extract a compact representation $T$ that maximizes the mean log likelihood ratio $\left\langle\log\left(\frac{p(y^+|t)}{p(y^-|t)}\right)\right\rangle_{p(t,y^+,y^-)}$, measuring the discriminability between the distribution of $p(y^+|t)$ and $p(y^-|t)$.

The above setup can be extended to the case of multiple variables on which multi-information should be preserved $\{y_1^+, ..., y_{N+}^+\}$ and variables on which multi-information should be removed $\{y_1^-, ..., y_{N-}^-\}$, as discussed in [8]. This yields

$$\frac{p(t|x)}{p(t)} \quad \propto \quad e^{\left(-\sum_i \gamma_i^+ D_{KL}\left[p(y_i^+|x)||p(y_i^+|t)\right] + \sum_i \gamma_i^- D_{KL}\left[p(y_i^-|x)||p(y_i^-|t)\right]\right)} \tag{6}$$

which can be solved together with the other self-consistent conditions, similarly to Eq. 4.

## 4  Relation to Rate Distortion Theory with Side Information

The problem formulated above is related to the theory of rate distortion with side information ([17],[2] p. 439). In rate distortion theory (RDT) a source variable $X$ is stochastically encoded into a variable $T$, which is decoded at the other side of the channel with some distortion. The achievable code rate, $R$ at a given distortion level $D$, is bounded by the optimal rate, also known as the *rate distortion function*, $R(D)$. The optimal encoding is determined by the stochastic map $p(t|x)$, where the representation quantization is found by minimizing the average distortion. For the optimal code $I(X;T) = R(D)$.

This rate can be improved by utilizing side information in the form of another variable, $W$, that is known at both ends of the channel. In this case, an improved rate can be achieved by avoiding sending information about $X$ that can be extracted from $W$. Indeed, in this case the rate distortion function with this side information has a lower lower-bound, given by $R(D) = I(X;T) - I(T;W)$, where $T$ is the optimal quantization of $X$ in this case, under the distortion constraint (see [17] for details). In the information bottleneck framework the average distortion is replaced by the mutual information about the relevant variable, while the rate-distortion function is turned into a convex curve that characterizes the complexity of the relation between the variables, (see [15, 13]).

Similarly, IBSI avoids differentiating instances of $X$ that are informative about $Y^+$ if they contain information also about $Y^-$. The variable $Y^-$ is analogous to the side information variable $W$, while $Y^+$ is just the "informative" $Y$ of the original IB. While the formal analogy between these problems helps in their mathematical formulation, it is important to emphasize that these are very different problems both in motivation and scope. Whereas RDT with side information is a specific communication problem with some given (often arbitrary) distortion function, our problem is a general statistical non-parametric analysis technique that depends solely by the choice of the variables $X$, $Y^+$ and $Y^-$. Many different pattern recognition and discriminative learning problems can be cast into this general information theoretic framework - far beyond the original setting of RDT with side information.

## 5  Algorithms

The set of self-consistent equations (Eq. 4), can be solved by iterating the equations, given initial distributions, similar to the algorithm presented for the IB [15, 8], with similar convergence proofs. Unlike the original IB equations, convergence of the algorithm is no longer allways guaranteed, simply because the problem is not guaranteed to have feasible solutions for all $\gamma$ values. However, there exist a non empty set of $\gamma$ values for which this algorithm is guaranteed to converge.

As in the case of IB, various heuristics can be applied, such as deterministic annealing - in which increasing the parameter $\beta$ is used to obtain finer clusters; greedy agglomerative hard clustering [13]; or a sequential K-means like algorithm [12]. The latter provides a good compromise between top-down annealing and agglomerative greedy approaches and

achieves excellent performance. This is the algorithm we adopted in this paper, modifying the algorithm detailed in [12], by using a target function $\mathcal{F} = \beta^{-1} I(T; X) - I(T; Y^+) + \gamma I(T; Y^-)$.

# 6  Applications

We describe two applications of our method: a simple synthetic example, and a "real world" problem of hierarchical text categorization. We also used IBSI to extract relevant features in face images, but these results will be published elsewhere due spavce considerations.

## 6.1  A synthetic example

To demonstrate the ability of our approach to uncover weak but interesting hidden structures in data, we designed a co-occurrences matrix contains two competing sub-structures (see figure 2A). For demonstration purposes, the matrix was created such that the stronger structure can be observed on the left and the weaker structure on the right. Compressing $X$ into two clusters while preserving information on $Y^+$ using IB ($\gamma = 0$), yields the clustering of figure 2B, in which the upper half of $x$'s are all clustered together. This clustering follows from the strong structure on the left of 2A.

We now created a second co-occurrence matrix, to be used for identifying the relevant structure, in which each half of $X$ yield similar distributions $P(y^-|x)$. Applying sequential-IBSI now successfully ignores the strong but irrelevant structure in $P(Y^+, X)$ and retrieves the weak structure. Importantly, this is done in an unsupervised manner, without explicitly pointing to the strong but irrelevant structure.

This example was designed for demonstration purposes, thus the irrelevant structures is strongly manifested in $\tilde{P}(X; Y^-)$. The next example shows that our approach is also useful for real data, in which structures are much more covert.

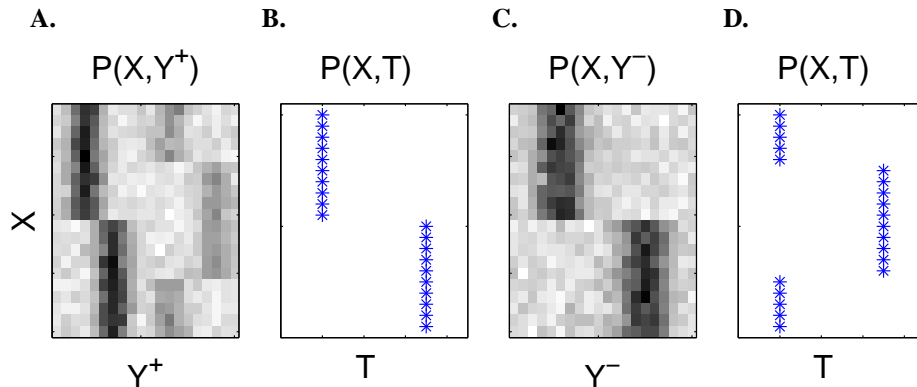

Figure 2: Demonstration of IBSI operation. **A.** A joint distribution $P(X, Y^+)$ that contains two distinct and conflicting structure. **B.** Clustering $X$ into two clusters using the information bottleneck method separates upper and lower values of $X$, according to the stronger structure. **C.** A joint distribution $P(X, Y^-)$ that contains a single structure, similar in nature to the stronger structure $P(X, Y^+)$. **D.** Clustering $X$ into two clusters using IBSI successfully extract the weaker structure in $P(X, Y^+)$.

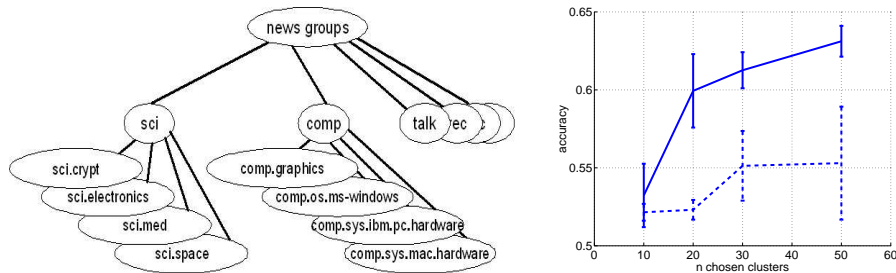

Figure 3: **A.** An illustration of the 20 newsgroups hierarchical data we used. **B.** Categorization accuracy vs. no of word clusters $k$. $N = 100$. IB dashed line. IBSI solid line.

## 6.2 Hierarchical text categorization

Text categorization is a fundamental task in information retrieval. Typically, one has to group a large set of texts into groups of homogeneous subjects. Recently, Slonim and colleagues showed that the IB method achieves categorization that predicts manually predefined categories with great accuracy, and largely outperforms competing methods [12]. Clearly, this unsupervised task becomes more difficult when the texts have similar subjects, because alternative categories are extracted instead of the "correct" one.

This problem can be alleviated by using side information in the form of additional documents from other categories. This is specifically useful in hierarchical document categorization, in which known categories are refined by grouping documents into sub-categories. [4, 16]. IBSI can be applied to this problem by operating on the terms-documents co-occurrence matrix while using the other top-level groups for focusing on the relevant structures. To this end, IBSI is used to identify clusters of terms that will be later used to cluster a group of documents into its subgroups,

While IBSI is targeted at learning structures in unsupervised manner, we have chosen to apply it to a labelled dataset of documents in order to be able to measure how its results agree with manual classification. Labels are not used by our algorithms during learning and serve only to quantify the performance. We used the *20 Newsgroups database* collected by [7] preprocessed as described in [12]. This database consists of 20 equal sized groups of documents, hierarchically organized into groups according to their content (figure 3A). We aimed to cluster documents that belong to two newsgroups from the supergroup of computer documents and have very similar subjects *comp.sys.ibm.pc.hardware* and *comp.sys.mac.hardware*. As side information we used all documents from the super group of science ( *sci.crypt*, *sci.electronics*, *sci.med*, *sci.space*).

To demonstrate the power of IBSI we used double clustering to separate documents into two groups. The goal of the first clustering phase is to use IBSI to identify clusters of terms that extract the relevant structures of the data. The goal of the second clustering phase is simply to provide a quantitative measure for the quality of the features extracted in the first phase. We therefor performed the following procedure: First, the most frequent 2000 words in these documents were clustered into $N$ clusters using IBSI. Then, word clusters were sorted by a single-cluster score $D_{KL}[p(y^+|t)\|p(y^+)] - \gamma D_{KL}[p(y^-|t)\|p(y^-)]$, and the $k$ clusters with the highest score were chosen. These word-clusters were then used for clustering documents. The performance of this process is evaluated by measuring the overlap of the resulting clusters with the manualy classified groups. Figure 3, plots document-clustering accuracy for $N = 100$, as a function of $k$. IBSI ($\gamma = 1$) is compared with the IB method (i.e. $\gamma = 0$). Using IBSI successfully improves mean clustering accuracy from about 55 percent to about 63 percents.

## 7  Discussion and Further Research

We have presented an information theoretic approach for extracting relevant structures from data, by utilizing additional data known to share irrelevant structures with the relevant data. Naturally, the choice of side data may considerably influence the solutions obtained with IBSI, simply because using different irrelevant variables, is equivalent to asking different questions about the data analysed. In practice, side data can be naturally defined in numerous applications, in particular in exploratory analysis of scientific experiments, e.g. when searching for features that characterize a disease but not healthy subjects.

While the current work is based on clustering to compress the source, the notion of extracting relevance through side information can be extended to other forms of dimentionality reduction, such as non-linear embedding on low dimensional manifolds. In particular side information can be naturally combined with information theoretic modeling approaches such as SDR [5]. Our preliminary results with this approach were found very promising.

## Acknowledgements

We thank Amir Globerson, Noam Slonim, Israel Nelken and Nir Friedman for helpful discussions. G.C. is supported by a grant from the ministry of Science, Israel.

## References

[1] L.D. Baker and A. K. McCallum. Distributional clustering of words for text classification. In *Proc. of SIGIR*, 1998.

[2] T.M. Cover and J.A. Thomas. *The elements of information theory*. Plenum Press, NY, 1991.

[3] I. Csiszar and J.Korner. *Information theory: Coding Theorems for Discrete Memoryless Systems*. Academic Press New York, 1997. 2nd edition.

[4] S. Dumais and H. Chen. Hierarchical classification of web content. In *Proc. of SIGIR*, pages 256–263, 2000.

[5] A. Globerson and N. Tishby. Sufficient dimentionality reduction. *J. Mach. Learn. Res.*, 2003.

[6] T. Hoffman. Probabilistic latent semantic indexing. In *Proc. of SIGIR*, pages 50–57, 1999.

[7] K. Lang. Learning to filter netnews. In *Proc. of 12th Int Conf. on machine Learning*, 1995.

[8] N. Friedman O. Mosenzon, N. Slonim, and N. Tishby. Multivariate information bottleneck. In *Proc of UAI*, pages 152–161, 2001.

[9] F.C. Pereira, N. Tishby, and L. Lee. Distributional clustering of english words. In *Meeting of the Association for Computational Linguistics*, pages 183–190, 1993.

[10] E. Schneidman, N. Slonim, N. Tishby, R. deRuyter van Steveninck, and W. Bialek. Analyzing neural codes using the information bottleneck method. Technical report, The Hebrew University, 2002.

[11] J. Sinkkonen and S. Kaski. Clustering based on conditional distribution in an auxiliary space. *Neural Computation*, 14:217–239, 2001.

[12] N. Slonim, N. Friedman, and N. Tishby. Unsupervised document classification using sequential information maximization. In *Proc. of SIGIR*, pages 129–136, 2002.

[13] N. Slonim and N. Tishby. Agglomerative information bottleneck. In *Advances in Neural Information Processing Systems (NIPS)*, 1999.

[14] N. Slonim and N. Tishby. Document clustering using word clusters via the information bottleneck method. In *Proc. of SIGIR*, pages 208–215, 2000.

[15] N. Tishby, F.C. Pereira, and W. Bialek. The information bottleneck method. In *Proc. of 37th Allerton Conference on communication and computation*, 1999.

[16] A. Vinokourov and M.Girolani. A probabilistic framework for the hierarchic organization and classification of document collections. *J. Intell. Info Sys.*, 18(23):153–172, 2002.

[17] A. Wyner and J. Ziv. The rate distortion function for source coding with side information at the decoder. *IEEE Trans. Information Theory*, 22(1):1–10, 1976.